# Networks for the Separation of Sources that are Superimposed and Delayed

John C. Platt          Federico Faggin
Synaptics, Inc.
2860 Zanker Road, Suite 206
San Jose, CA 95134

## ABSTRACT

We have created new networks to unmix signals which have been mixed either with time delays or via filtering. We first show that a subset of the Hérault-Jutten learning rules fulfills a principle of minimum output power. We then apply this principle to extensions of the Hérault-Jutten network which have delays in the feedback path. Our networks perform well on real speech and music signals that have been mixed using time delays or filtering.

## 1  INTRODUCTION

Recently, there has been much interest in neural architectures to solve the "blind separation of signals" problem (Hérault & Jutten, 1986) (Vittoz & Arreguit, 1989). The separation is called "blind," because nothing is assumed known about the frequency or phase of the signals.

A concrete example of blind separation of sources is when the pure signals are sounds generated in a room and the mixed signals are the output of some microphones. The mixture process would model the delay of the sound to each microphone, and the mixing of the sounds at each microphone. The inputs to the neural network would be the microphone outputs, and the neural network would try to produce the pure signals.

The mixing process can take on different mathematical forms in different situations. To express these forms, we denote the pure signal $i$ as $P_i$, the mixed signal $i$ as $I_i$ (which is the $i$th input to the network), and the output signal $i$ as $O_i$.

The simplest form to unmix is linear superposition:

$$I_i(t) = P_i(t) + \sum_{j \neq i} M_{ij}(t)P_j(t). \tag{1}$$

A more realistic, but more difficult form to unmix is superposition with single delays:

$$I_i(t) = P_i(t) + \sum_{j \neq i} M_{ij}(t) P_j(t - D_{ij}(t)). \qquad (2)$$

Finally, a rather general mixing process would be superposition with causal filtering:

$$I_i(t) = P_i(t) + \sum_{j \neq i} \sum_k M_{ijk}(t) P_j(t - \delta_k). \qquad (3)$$

Blind separation is interesting for many different reasons. The network must adapt on-line and without a supervisor, which is a challenging type of learning. One could imagine using a blind separation network to clean up an input to a speech understanding system. (Jutten & Hérault, 1991) uses a blind separation network to deskew images. Finally, researchers have implemented blind separation networks using analog VLSI to yield systems which are capable of performing the separation of sources in real time (Vittoz & Arreguit, 1990) (Cohen, et. al., 1992).

## 1.1   Previous Work

Interest in adaptive systems which perform noise cancellation dates back to the 1960s and 1970s (Widrow, et. al., 1975). The first neural network to unmix on-line a linear superposition of sources was (Hérault & Jutten, 1986). Further work on off-line blind separation was performed by (Cardoso, 1989). Recently, a network to unmix filtered signals was proposed in (Jutten, et. al., 1991), independently of this paper.

# 2   PRINCIPLE OF MINIMUM OUTPUT POWER

In this section, we apply the mathematics of noise-cancelling networks (Widrow, et. al., 1975) to the network in (Hérault & Jutten, 1986) in order to generalize to new networks that can handle delays in the mixing process.

## 2.1   Noise-cancellation Networks

A noise-cancellation network tries to purify a signal which is corrupted by filtered noise (Widrow, et. al., 1975). The network has access to the isolated noise signal. The interference equation is

$$I(t) = P(t) + \sum_j M_j N(t - \delta_j). \qquad (4)$$

The adaptive filter inverts the interference equation, to yield an output:

$$O(t) = I(t) - \sum_j C_j N(t - \delta_j). \qquad (5)$$

The adaptation of a noise-cancellation network relies on an elegant notion: if a signal is impure, it will have a higher power than a pure signal, because the noise power adds to the signal power. The true pure signal has the lowest power. This minimum output power principle is used to determine adaptation laws for noise-cancellation networks. Specifically, at any time $t$, $C_j$ is adjusted by taking a step that minimizes $O(t)^2$

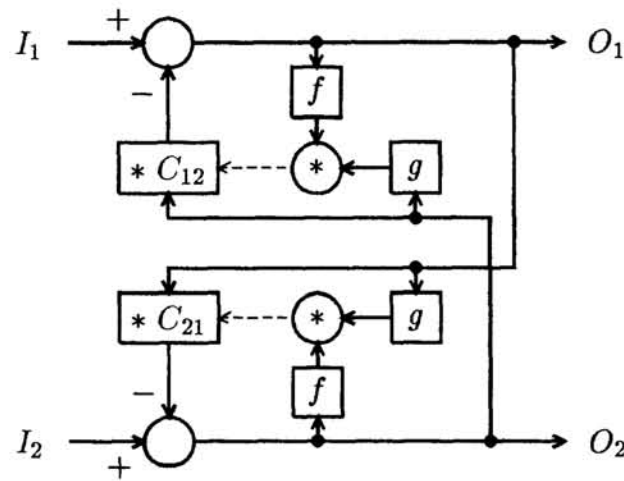

**Figure 1:** The network described in (Hérault & Jutten, 1986). The dashed arrows represent adaptation.

## 2.2   The Hérault-Jutten Network

The Hérault-Jutten network (see Figure 1) uses a purely additive model of interference. The interference is modeled by

$$I_i = P_i + \sum_{j \neq i} M_{ij} P_j. \tag{6}$$

Notice the Hérault-Jutten network solves a more general problem than previous noise-cancellation networks: the Hérault-Jutten network has no access to any pure signal.

In (Hérault & Jutten, 1986), the authors also propose inverting the interference model:

$$O_i = I_i - \sum_{j \neq i} C_{ij} O_j. \tag{7}$$

The Hérault-Jutten network can be understood intuitively by assuming that the network has already adapted so that the outputs are the pure signals ($O_j = P_j$). Each connection $C_{ij}$ subtracts just the right amount of the pure signal $P_j$ from the input $I_i$ to yield the pure signal $P_i$. So, the Hérault-Jutten network will produce pure signals if the $C_{ij} = M_{ij}$.

In (Hérault & Jutten, 1986), the authors propose a very general adaptation rule for the $C_{ij}$:

$$\Delta C_{ij}(t) = \eta f(O_i(t)) g(O_j(t)). \tag{8}$$

for some non-linear functions $f$ and $g$. (Sorouchyari, 1991) proves that the network converges for $f(x) = x^3$.

In this paper, we propose that the same elegant minimization principle that governs the noise-cancellation networks can be used to justify a subset of Hérault-Jutten

learning algorithms. Let $g(x) = x$ and $f(x)$ be a derivative of some convex function $h(x)$, with a minimum at $x = 0$. In this case, each output of the Hérault-Jutten network *independently* minimizes a function $h(x)$.

A Hérault-Jutten network can be made by setting $h(x) = x^2$. Unfortunately, this network will not converge, because the update rules for two connections $C_{ij}$ and $C_{ji}$ are identical:

$$\Delta C_{ij}(t) = \Delta C_{ji}(t) = O_i(t)O_j(t). \tag{9}$$

Under this condition, the two parameters $C_{ij}$ and $C_{ji}$ will track one another and not converge to the correct answer. Therefore, a non-linear adaptation rule is needed to break the symmetry between the outputs.

The next two sections of the paper describe how the minimum output power principle can be applied to generalizations of the Hérault-Jutten architecture.

## 3    NETWORK FOR UNMIXING DELAYED SIGNALS

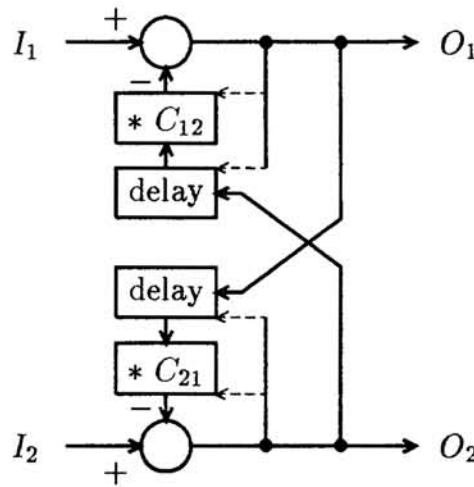

**Figure 2:**    Our network for unmixing signals mixed with single delays. The adjustable delay in the feedback path avoids the degeneracy in the learning rule. The dashed arrows represent adaptation: the source of the arrow is the source of the error used by gradient descent.

Our new network is an extension of the Hérault-Jutten network (see Figure 2). We assume that the interference is delayed by a certain amount:

$$I_i(t) = P_i(t) + \sum_{i \neq j} M_{ij} P_j(t - D_{ij}(t)). \tag{10}$$

Compare this to equation (6): our network can handle delayed interference, while the Hérault-Jutten network cannot. We introduce an adjustable delay in the feedback path in order to cancel the delay of the interference:

$$O_i(t) = I(t) - \sum_{i \neq j} C_{ij} O_j(t - d_{ij}(t)). \tag{11}$$

We apply the minimum output power principle to adapt the mixing coefficients $C_{ij}$ and the delays $d_{ij}$:

$$\Delta C_{ij}(t) = \alpha O_i(t)O_j(t - d_{ij}(t)),$$
$$\Delta d_{ij}(t) = -\beta C_{ij}(t)O_i(t)\frac{dO_j}{dt}(t - d_{ij}(t)).$$

(12)

By introducing a delay in the feedback, we prevent degeneracy in the learning rule, hence we can use a quadratic power to adjust the coefficients.

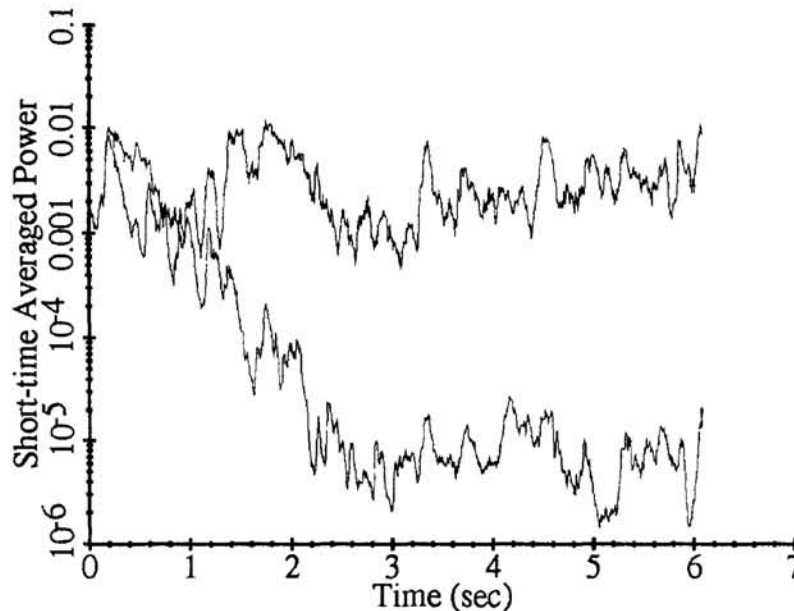

**Figure 3:** The results of the network applied to a speech/music superposition. These curves are short-time averages of the power of signals. The upper curve shows the power of the pure speech signal. The lower curve shows the power of the difference between the speech output of the network, and the pure speech signal. The gap between the curves is the amount that the network attenuates the interference between the music and speech: the adaptation of the network tries to drive the lower curve to zero. As you can see, the network quickly isolates the pure speech signal.

For a test of our network, we took two signals, one speech and one music, and mixed them together via software to form two new signals: the first being speech plus a delayed, attenuated music; the second being music plus delayed, attenuated speech. Figure 3 shows the results of our network applied to these two signals: the interference was attenuated by approximately 22 dB. One output of the network sounds like speech, with superimposed music which quickly fades away. The other output of the network sounds like music, with a superimposed speech signal which quickly fades away.

Our network can also be extended to more than two sources, like the Hérault-Jutten network. If the network tries to separate $S$ sources, it requires $S$ non-identical

inputs. Each output connects to one input, and a delayed version of each of the other outputs, for a total of $2S(S-1)$ adaptive coefficients.

## 4    NETWORK FOR UNMIXING FILTERED SIGNALS

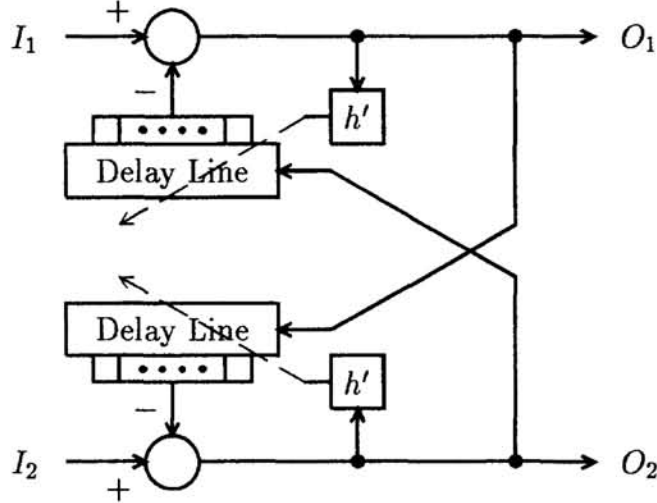

**Figure 4:**   A network to unmix signals that have been mixed via filtering. The filters in the feedback path are adjusted to independently minimize the power $h(O_i)$ of each output.

For the mixing process that involves filtering,

$$I_i(t) = P_i(t) + \sum_{j \neq i} \sum_k M_{ijk} P_j(t - \delta_k), \tag{13}$$

we put filters in the feedback path of each output:

$$O_i(t) = I_i(t) - \sum_{j \neq i} \sum_k C_{ijk} O_j(t - \delta_k). \tag{14}$$

(Jutten, et. al., 1991) also independently developed this architecture. We can use the principle of minimum output power to develop a learning rule for this architecture:

$$\Delta C_{ijk} = \eta h'(O_i(t))O_j(t - \delta_k) \tag{15}$$

for some convex function $h$. (Jutten, et. al., 1991) suggests using an adaptation rule that is equivalent to choosing $h(x) = x^4$.

Interestingly, neither the choice of $h(x) = x^2$ nor $h(x) = x^4$ converges to the correct solution. For both $h(x) = x^2$ and $h(x) = x^4$, if the coefficients start at the correct solution, they stay there. However, if the coefficients start at zero, they converge to a solution that is only roughly correct (see Figure 5). These experiments show

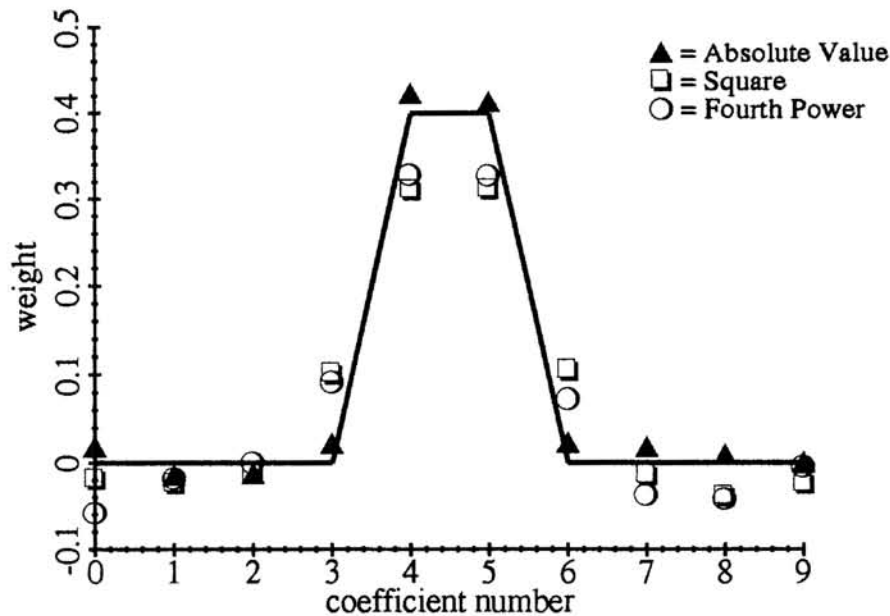

**Figure 5:** The coefficients for one filter in the feedback path of the network. The weights were initialized to zero. Two different speech/music mixtures were applied to the network. The solid line indicates the correct solution for the coefficients. When minimizing either $h(x) = x^2$ or $h(x) = x^4$, the network converges to an incorrect solution. Minimizing $h(x) = |x|$ seems to work well.

that the learning algorithm has multiple stable states. Experimentally, the spurious stable states seem to perform roughly as well as the true answer.

To account for these multiple stable states, we came up with a conjecture: that the different minimizations performed by each output fought against one another and created the multiple stable states. Optimization theory suggests using an exact penalty method to avoid fighting between multiple terms in a single optimization criteria (Gill, 1981). The exact penalty method minimizes a function $h(x)$ that has a non-zero derivative for $x$ close to 0. We tried a simple exact penalty method of $h(x) = |x|$, and it empirically converged to the correct solution (see Figure 5). The adaptation rule is then

$$\Delta C_{ijk} = \eta \, \text{sgn}(O_i(t)) O_j(t - \delta_k) \tag{16}$$

In this case, the non-linearity of the adaptation rule seems to be important for the network to converge to the true answer. For a speech/music mixture, we achieved a signal-to-noise ratio of 20 dB using the update rule (16).

## 5    FUTURE WORK

The networks described in the last two sections were found to converge empirically. In the future, proving conditions for convergence would be useful. There are some known pathological cases which cause these networks not to converge. For example, using white noise as the pure signals for the network in section 3 causes it to fail, because there is no sensible way for the network to change the delays.

More exploration of the choice of optimization function needs to be performed in the future. The work in section 4 is just a first step which illustrates the possible usefulness of the absolute value function.

Another avenue of future work is to try to express the blind separation problem as a global optimization problem, perhaps by trying to minimize the mutual information between the outputs. (Feinstein, Becker, personal communication)

# 6   CONCLUSIONS

We have found that the minimum output power principle can generate a subset of the Hérault-Jutten network learning rules. We use this principle to adapt extensions of the Hérault-Jutten network, which have delays in the feedback path. These new networks unmix signals which have been mixed with single delays or via filtering.

**Acknowledgements**

We would like to thank Kannan Parthasarathy for his assistance in some of the experiments. We would also like to thank David Feinstein, Sue Becker, and David Mackay for useful discussions.

**References**

Cardoso, J. F., (1989) "Blind Identification of Independent Components," *Proceedings of the Workshop on Higher-Order Spectral Analysis*, Vail, Colorado, pp. 157–160, (1989).

Cohen, M. H., Pouliquen, P. O., Andreou, A. G., (1992) "Analog VLSI Implementation of an Auto-Adaptive Network for Real-Time Separation of Independent Signals," *Advances in Neural Information Processing Systems 4*, Morgan-Kaufmann, San Mateo, CA.

Gill, P. E., Murray, W., Wright, M. H., (1981) *Practical Optimization*, Academic Press, London.

Herault, J., Jutten, C., (1986) "Space or Time Adaptive Signal Processing by Neural Network Models," *Neural Networks for Computing*, AIP Conference Proceedings 151, pp. 207–211, Snowbird, Utah.

Jutten, C., Thi, L. N., Dijkstra, E., Vittoz, E., Caelen, J., (1991) "Blind Separation of Sources: an Algorithm for Separation of Convolutive Mixtures," *Proc. Intl. Workshop on High Order Statistics*, Chamrousse France, July 1991.

Jutten, C., Herault, J., (1991) "Blind Separation of Sources, part I: An Adaptive Algorithm Based on Neuromimetic Architecture," *Signal Processing*, vol. 24, pp. 1–10.

Sorouchyari, E., (1991) "Blind Separation of Sources, Part III: Stability analysis," *Signal Processing*, vol. 24, pp. 21–29.

Vittoz, E. A., Arreguit, X., (1989) "CMOS Integration of Herault-Jutten Cells for Separation of Sources," *Proc. Workshop on Analog VLSI and Neural Systems*, Portland, Oregon, May 1989.

Widrow, B., Glover, J., McCool, J., Kaunitz, J., Williams, C., Hearn, R., Zeidler, J., Dong, E., Goodlin, R., (1975) "Adaptive Noise Cancelling: Principles and Applications," *Proc. IEEE*, vol. 63, no. 12, pp. 1692–1716.
